# Scrambled Objects for Least-Squares Regression

**Odalric-Ambrym Maillard** and **Rémi Munos**
SequeL Project, INRIA Lille - Nord Europe, France
`{odalric.maillard, remi.munos}@inria.fr`

## Abstract

We consider least-squares regression using a randomly generated subspace $\mathcal{G}_P \subset \mathcal{F}$ of finite dimension $P$, where $\mathcal{F}$ is a function space of infinite dimension, e.g. $L_2([0,1]^d)$. $\mathcal{G}_P$ is defined as the span of $P$ random features that are linear combinations of the basis functions of $\mathcal{F}$ weighted by random Gaussian i.i.d. coefficients. In particular, we consider multi-resolution random combinations at all scales of a given mother function, such as a hat function or a wavelet. In this latter case, the resulting Gaussian objects are called *scrambled wavelets* and we show that they enable to approximate functions in Sobolev spaces $H^s([0,1]^d)$. As a result, given $N$ data, the least-squares estimate $\widehat{g}$ built from $P$ scrambled wavelets has excess risk $||f^* - \widehat{g}||_{\mathcal{P}}^2 = O(||f^*||_{H^s([0,1]^d)}^2 (\log N)/P + P(\log N)/N)$ for target functions $f^* \in H^s([0,1]^d)$ of smoothness order $s > d/2$. An interesting aspect of the resulting bounds is that they do not depend on the distribution $\mathcal{P}$ from which the data are generated, which is important in a statistical regression setting considered here. Randomization enables to adapt to any possible distribution.

We conclude by describing an efficient numerical implementation using lazy expansions with numerical complexity $\tilde{O}(2^d N^{3/2} \log N + N^2)$, where $d$ is the dimension of the input space.

## 1 Introduction

We consider ordinary least-squares regression using randomly generated feature spaces. Let us first describe the general regression problem: we observe data $\mathcal{D}_N = (\{x_n, y_n\}_{1 \leq n \leq N})$ (with $x_n \in \mathcal{X}$ a compact subset of $\mathbb{R}^d$, and $y_n \in \mathbb{R}$), assumed to be independently and identically distributed (i.i.d.) with $x_n \sim \mathcal{P}$ and

$$y_n = f^*(x_n) + \eta_n,$$

where $f^*$ is the (unknown) target function, such that $||f^*||_\infty \leq L$, and $\eta_n$ is a centered, independent noise of variance bounded by $\sigma^2$. We assume that $L$ and $\sigma$ are known.

Now, for a given class of functions $\mathcal{F}$, and $f \in \mathcal{F}$, we define the empirical $\ell_2$-error

$$L_N(f) \stackrel{\text{def}}{=} \frac{1}{N} \sum_{n=1}^{N} [y_n - f(x_n)]^2,$$

and the generalization error

$$L(f) \stackrel{\text{def}}{=} \mathbb{E}_{X,Y}[(Y - f(X))^2].$$

The goal is to return a regression function $\widehat{f} \in \mathcal{F}$ with lowest possible generalization error $L(\widehat{f})$. The excess risk $L(\widehat{f}) - L(f^*) = ||f^* - \widehat{f}||_{\mathcal{P}}$ (where $||g||_{\mathcal{P}}^2 = \mathbb{E}_{X \sim \mathcal{P}}[g(X)^2]$) measures the closeness to optimality.

In this paper we consider infinite dimensional spaces $\mathcal{F}$ that are generated by a denumerable family of functions $\{\varphi_i\}_{i \geq 1}$, called *initial features* (such as wavelets). We will assume that $f^* \in \mathcal{F}$.

Since $\mathcal{F}$ is an infinite dimensional space, the empirical risk minimizer in $\mathcal{F}$ is certainly subject to overfitting. Traditional methods to circumvent this problem have considered penalization, i.e. one searches for a function in $\mathcal{F}$ which minimizes the empirical error plus a penalty term, for example $\widehat{f} = \arg\min_{f \in \mathcal{F}} L_N(f) + \lambda ||f||_p^p$ for $p = 1$ or $2$, where $\lambda$ is a parameter and usual choices for the norm are $\ell_2$ (ridge-regression [17]) and $\ell_1$ (LASSO [16]).

In this paper we follow an alternative approach introduced in [10], called Compressed Least Squares Regression, which considers generating randomly a subspace $\mathcal{G}_P$ (of finite dimension $P$) of $\mathcal{F}$, and then returning the empirical risk minimizer in $\mathcal{G}_P$, i.e. $\arg\min_{g \in \mathcal{G}_P} L_N(g)$. This previous work considered the case when $\mathcal{F}$ is of finite dimension. Here we consider specific cases of infinite dimensional spaces $\mathcal{F}$ and provide a characterization of the resulting approximation spaces.

## 2 Regression with random spaces

Let us briefly recall the method described in [10] and extend it to the case of infinite dimensional spaces $\mathcal{F}$. In this paper we assume that the set of features $(\varphi_i)_{i \geq 1}$ are continuous and are such that,

$$\sup_{x \in \mathcal{X}} ||\varphi(x)||^2 < \infty, \text{ where } ||\varphi(x)||^2 \stackrel{\text{def}}{=} \sum_{i \geq 1} \varphi_i(x)^2. \tag{1}$$

Examples of feature spaces satisfying this property include rescaled wavelets and will be described in Section 3.

The random subspace $\mathcal{G}_P$ is generated by building a set of $P$ *random features* $(\psi_p)_{1 \leq p \leq P}$ defined as linear combinations of the initial features $\{\varphi_i\}_{1 \geq 1}$ weighted by random coefficients:

$$\psi_p(x) \stackrel{\text{def}}{=} \sum_{i \geq 1} A_{p,i} \varphi_i(x), \text{ for } 1 \leq p \leq P, \tag{2}$$

where the (infinitely many) coefficients $A_{p,i}$ are drawn i.i.d. from a centered distribution with variance $1/P$. Here we explicitly choose a Gaussian distribution $\mathcal{N}(0, 1/P)$. Such a definition of the features $\psi_p$ as an infinite sum of random variable is not obvious (this is called an expansion of a Gaussian object) and we refer to [11] for elements of theory about Gaussian objects and for the expansion of a Gaussian object. It is shown that under assumption (1), the random features are well defined. Actually, they are random samples of a centered Gaussian process indexed by the space $\mathcal{X}$ with covariance structure given by $\frac{1}{P} \langle \varphi(x), \varphi(x') \rangle$, where we use the notation $\langle u, v \rangle = \sum_i u_i v_i$ for two square-summable sequences $u$ and $v$. Indeed, $\mathbb{E}_{A_p}[\psi_p(x)] = 0$, and

$$\text{Cov}_{A_p}(\psi_p(x), \psi_p(x')) = \mathbb{E}_{A_p}[\psi_p(x)\psi_p(x')] = \frac{1}{P} \sum_{i \geq 1} \varphi_i(x)\varphi_i(x') = \frac{1}{P} \langle \varphi(x), \varphi(x') \rangle.$$

The continuity of the initial features $(\varphi_i)$ guarantees that there exists a continuous version of the process $\psi_p$ which is thus a Gaussian process.

Then we define $\mathcal{G}_P \subset \mathcal{F}$ to be the (random) vector space spanned by those features, i.e.

$$\mathcal{G}_P \stackrel{\text{def}}{=} \{g_\beta(x) \stackrel{\text{def}}{=} \sum_{p=1}^P \beta_p \psi_p(x), \beta \in \mathbb{R}^P\}.$$

Now, the least-squares estimate $g_{\widehat{\beta}} \in \mathcal{G}_P$ is the function in $\mathcal{G}_P$ with minimal empirical error, i.e.

$$g_{\widehat{\beta}} = \arg\min_{g_\beta \in \mathcal{G}_P} L_N(g_\beta), \tag{3}$$

and is the solution of a least-squares regression problem, i.e. $\widehat{\beta} = \Psi^\dagger Y \in \mathbb{R}^P$, where $\Psi$ is the $N \times P$-matrix composed of the elements: $\Psi_{n,p} \stackrel{\text{def}}{=} \Psi_p(x_n)$, and $\Psi^\dagger$ is the Moore-Penrose pseudo-inverse of $\Psi$[1]. The final prediction function $\widehat{g}(x)$ is the truncation (to the threshold $\pm L$) of $g_{\widehat{\beta}}$, i.e.

$\widehat{g}(x) \stackrel{\text{def}}{=} T_L[g_{\widehat{\beta}}(x)]$, where $T_L(u) \stackrel{\text{def}}{=} \begin{cases} u & \text{if } |u| \leq L, \\ L \, \text{sign}(u) & \text{otherwise.} \end{cases}$

Next, we provide bounds on the approximation error of $f^*$ in $\mathcal{G}_P$ and deduce excess risk bounds.

## 2.1 Approximation error

We now extend the result of [10] and derive approximation error bounds both in expectation and in high probability. We restrict the set of target functions to belong to the approximation space $\mathcal{K} \subset \mathcal{F}$ (also identified to the kernel space associated to the expansion of a Gaussian object):

$$\mathcal{K} \overset{\text{def}}{=} \{ f_\alpha \in \mathcal{F}, ||\alpha||^2 \overset{\text{def}}{=} \sum_{i \geq 1} \alpha_i^2 < \infty \}. \tag{4}$$

**Remark 1.** *This space may be seen from two equivalent points of view: either as a set of functions that are random linear combinations of the initial features, or a set of functions that are the expectation of some random processes (interpretation in terms of kernel space). We will not develop the related theory of Gaussian processes here but we refer the reader interested in the construction of kernel spaces to [11]*

Let $f_\alpha = \sum_i \alpha_i \varphi_i \in \mathcal{K}$. Write $g^*$ the projection of $f_\alpha$ onto $\mathcal{G}_P$ w.r.t. the norm $|| \cdot ||_{\mathcal{P}}$, i.e. $g^* = \arg\min_{g \in \mathcal{G}_P} ||f_\alpha - g||_{\mathcal{P}}$, and $\bar{g}^* = T_L g^*$ its truncation at the threshold $L \geq ||f_\alpha||_\infty$. Notice that due to the randomness of the features $(\psi_p)_{1 \leq p \leq P}$ of $\mathcal{G}_P$, the space $\mathcal{G}_P$ is also random, and so is $\bar{g}^*$. The following result provides bounds for the approximation error $||f_\alpha - \bar{g}^*||_{\mathcal{P}}$ both in expectation and in high probability.

**Theorem 1.** *For any $\eta > 0$, whenever $P \geq c_1 \log(P\gamma^2 \sqrt{\log(1/\eta)/\eta})$, we have with probability $1 - \eta$ (w.r.t. the choice of the random subspace $\mathcal{G}_P$),*

$$\inf_{g \in \mathcal{G}} ||f^* - T_L(g)||_{\mathcal{P}}^2 \leq c_2 \frac{||\alpha||^2 \sup_x ||\varphi(x)||^2}{P} \big( 1 + \log(P\gamma^2 \sqrt{\log(1/\eta)/\eta}) \big),$$

*where $\gamma = \frac{L}{||\alpha|| \sup_x ||\varphi(x)||}$ and $c_1, c_2$ are some universal constants (see [11]). A similar result holds in expectation.*

This result relies on the property that $\inf_{g \in \mathcal{G}_P} ||f_\alpha - g||_{\mathcal{P}} \leq ||f_\alpha - g_{A\alpha}||_{\mathcal{P}}$ and that $g_{A\alpha}$, considered as a random variable w.r.t. the choice of the random elements $A$, concentrates around $f_\alpha$ (in $|| \cdot ||_{\mathcal{P}}$-norm) when $P$ increases. Indeed, $g_{A\alpha}(x) = (A\alpha) \cdot \psi(x) = (A\alpha) \cdot (A\varphi(x))$ which is close to $\alpha \cdot \varphi(x) = f_\alpha(s)$, since inner-products are approximately preserved through random projections (from a variant of Johnson-Lindenstrauss (JL) Lemma). The proof of Theorem 1 (provided in Appendix of [11]) relies in generating auxiliary samples $X_1', \dots, X_J'$ from $\mathcal{P}$, applying JL Lemma at those points and combining it with a Chernoff-Hoeffding bound for generalizing the result to hold in $|| \cdot ||_{\mathcal{P}}$-norm.

**Remark 2.** *An interesting property of this result is that the bound does not depend on the distribution $\mathcal{P}$. This distribution is used in the definition of the norm $|| \cdot ||_{\mathcal{P}}$ to assess how well a function space $\mathcal{G}_P$ can approximate a function $f_\alpha$. It is thus surprising that the measure $\mathcal{P}$ does not appear in the bound. Actually, the fact that $\mathcal{G}_P$ is random enables it to be close to $f_\alpha$ (in high probability or in expectation) whatever the measure $\mathcal{P}$ is. This is especially interesting in a regression setting where the distribution $\mathcal{P}$ from which the data are generated is not known in advance.*

## 2.2 Excess risk bounds

We now combine the approximation error bound from Theorem 1 with usual estimation error bounds for linear spaces (see e.g. [7]). Let us consider a target function $f^* = \sum_i \alpha_i^* \varphi_i \in \mathcal{K}$. Remember that our prediction function $\widehat{g}$ is the truncation $\widehat{g} \overset{\text{def}}{=} T_L[g_{\widehat{\beta}}]$ of the (ordinary) least-squares estimate $g_{\widehat{\beta}}$ (empirical risk minimizer in the random space $\mathcal{G}_P$) defined by (3).

We now provide upper bounds (both in expectation and in high probability) on the excess risk for the least-squares estimate using random subspaces (the proof is given in [11]).

**Theorem 2.** *Whenever $P \geq c_3 \log N$, we have the following bound in expectation (w.r.t. all sources of randomness, i.e. input data, noise, and the choice of the random features):*

$$\mathbb{E}_{G_P, X, Y} ||f^* - \widehat{g}||_{\mathcal{P}}^2 \leq c_4 \big( \sigma^2 \frac{P}{N} + L^2 \frac{P \log N}{N} + \frac{\log N}{P} ||\alpha^*||^2 \sup_x ||\varphi(x)||^2 \big), \tag{5}$$

*Now, for any $\eta > 0$, whenever $P \geq c_5 \log(N/\eta)$, we have the following bound in high probability (w.r.t. the choice of the random features), where $c_3, c_4, c_5, c_6$ are universal constant (see [11]):*

$$\mathbb{E}_{X, Y} ||f^* - \widehat{g}||_{\mathcal{P}}^2 \leq c_6 \big( \sigma^2 \frac{P}{N} + L^2 \frac{P \log N}{N} + \frac{\log N/\eta}{P} ||\alpha^*||^2 \sup_x ||\varphi(x)||^2 \big). \tag{6}$$

The results of Theorems 1 and 2 say that if the term $||\alpha^*||^2 \sup_x ||\varphi(x)||^2$ is small, then the least-squares estimate in the random subspace $\mathcal{G}_P$ has low excess risk. The question we wish to address now is whether we can define spaces for which this is the case. In the next section we provide two examples of feature spaces and characterize the space of functions for which this term is controlled.

## 3  Regression with Scrambled Objects

In the two examples provided below we consider (infinitely many) initial features that are translations and rescaling of a given mother function (which is assumed to be continuous) at all scales. Thus each random feature $\psi_p$ is a Gaussian object based on a multi-scale scheme built from an object (the mother function), and will be called a "scrambled object", to refer to the disorderly construction of this multi-resolution random process.

We thus propose to solve the regression problem by ordinary Least Squares on the (random) approximation space defined by the span of $P$ such scrambled objects. In the next sections we provide two examples. The first one considers the case when the mother function is a hat function and we show that the corresponding scrambled objects are Brownian motions. The second example considers wavelets. The proof of bounds (7) and (8) can be found in [11].

### 3.1  Brownian motions and Brownian Sheets

**Dimension** 1**:**  We start with the 1-dimensional case where $\mathcal{X} = [0,1]$. Let us choose as object (mother function) the hat function $\Lambda(x) = x\mathbb{I}_{[0,1/2[} + (1-x)\mathbb{I}_{[1/2,1[}$. We define the (infinite) set of initial features as translated and rescaled hat functions: $\Lambda_{j,l}(x) = 2^{-j/2}\Lambda(2^j x - l)$ for any scale $j \geq 1$ and translation index $0 \leq l \leq 2^j - 1$. We also write $\Lambda_{0,0}(x) = x$. This defines a basis of the space of continuous functions $\mathcal{C}^0([0,1])$ equal to 0 at 0 (introduced by Faber in 1910, and known as the Schauder basis, see [8] for an interesting overview). Those functions are indexed by the scale $j$ and translation index $l$, but all functions may be equivalently indexed by a unique index $i \geq 1$.

We have the property that the random features $\psi_p(x)$, defined as linear combinations of those hat functions weighted by Gaussian i.i.d. random numbers, are Brownian motions (See Example 1 of [11] for the proof). In addition, we can characterize the corresponding kernel space $\mathcal{K}$, which is the Sobolev space $H^1([0,1])$ of order 1 (space of functions which have a weak derivative in $L_2([0,1])$).

**Dimension** $d$**:**  For the extension to dimension $d$, we define the initial features as the tensor product $\varphi_{j,l}$ of one-dimensional hat functions (thus $j$ and $l$ are multi-indices). The random features $\psi_p(x)$ are Brownian sheets (extensions of Brownian motions to several dimensions) and the corresponding kernel $\mathcal{K}$ is the so-called *Cameron-Martin space* [9], endowed with the norm $||f||_{\mathcal{K}} = ||\frac{\partial^d f}{\partial x_1 \ldots \partial x_d}||_{L^2([0,1]^d)}$ (see also Example 1 of [11] for the proof). One may interpret this space as the set of functions which have a $d$-th order crossed (weak) derivative $\frac{\partial^d f}{\partial x_1 \ldots \partial x_d}$ in $L^2([0,1]^d)$, vanishing on the "left" boundary (edges containing 0) of the unit $d$-dimensional cube. Note that in dimension $d > 1$, this space differs from the Sobolev space $H^1$.

**Regression with Brownian Sheets:**  When one uses Brownian sheets for regression with a target function $f^* = \sum_i \alpha_i^* \varphi_i$ that lies in the Cameron-Martin space $\mathcal{K}$ defined previously (i.e. such that $||\alpha^*|| < \infty$), then the term $||\alpha^*||^2 \sup_{x \in \mathcal{X}} ||\varphi(x)||^2$ that appears in Theorems 1 and 2 is bounded as:

$$||\alpha^*||^2 \sup_{x \in \mathcal{X}} ||\varphi(x)||^2 \leq 2^{-d}||f^*||_{\mathcal{K}}^2.$$

Thus, from Theorem 2, ordinary least-squares performed on random subspaces spanned by $P$ Brownian sheets has an expected excess risk

$$\mathbb{E}_{G_P, X, Y}||f^* - \widehat{g}||_{\mathcal{P}}^2 = O\Big(\frac{\log N}{N}P + \frac{\log N}{P}||f^*||_{\mathcal{K}}^2\Big), \tag{7}$$

(and a similar bound holds in high probability).

## 3.2 Scrambled Wavelets in $[0,1]^d$

We now introduce a second example built from a family of orthogonal wavelets $(\tilde{\varphi}_{\varepsilon,j,l}) \in C^q([0,1]^d)$ (where $\varepsilon \in \{0,1\}^d$ is a multi-index, $j$ is a scale index, $l$ a multi-index, see [2, 12] for details of the notations) with at least $q > d/2$ vanishing moments. Now for $s \in (d/2, q)$, we define the initial features $(\varphi_{\varepsilon,j,l})$ as the rescaled wavelets $(\tilde{\varphi}_{\varepsilon,j,l})$, i.e. $\varphi_{\varepsilon,j,l} \stackrel{\text{def}}{=} 2^{-js} \frac{\tilde{\varphi}_{\varepsilon,j,l}}{||\tilde{\varphi}_{\varepsilon,j,l}||_2}$. Again, the initial features may equivalently be indexed by a unique index $i \geq 1$. The random features $\psi_p$ defined from (2) are called "scrambled wavelets". It can be shown that the resulting approximation space $\mathcal{K}$ (i.e. $\{f_\alpha = \sum_i \alpha_i \varphi_i, ||\alpha|| < \infty\}$) is the Sobolev space $H^s([0,1]^d)$.

**Regression with Scambled Wavelets:** Assume that the mother wavelet $\tilde{\varphi}$ has compact support $[0,1]^d$ and is bounded by $\lambda$, and assume that the target function $f^* = \sum_i \alpha_i^* \varphi_i$ lies in the Sobolev space $H^s([0,1]^d)$ with $s > d/2$ (i.e. such that $||\alpha^*|| < \infty$). Then, we have,

$$||\alpha^*||^2 \sup_{x \in \mathcal{X}} ||\varphi(x)||^2 \leq \frac{\lambda^{2d}(2^d - 1)}{1 - 2^{-2(s-d/2)}} ||f^*||^2_{H^s([0,1]^d)}.$$

Thus from Theorem 2, ordinary least-squares performed on random subspaces spanned by $P$ scrambled wavelets has an expected excess risk

$$\mathbb{E}_{\mathcal{G}_P, X, Y}||f^* - \widehat{g}||^2_{\mathcal{P}} = O\Big(\frac{\log N}{N}P + \frac{\log N}{P}||f^*||^2_{H^s([0,1]^d)}\Big), \tag{8}$$

(and a similar bound holds in high probability).

In both examples, by choosing $P$ of order $\sqrt{N}||f^*||_{\mathcal{K}}$, one deduces the excess risk

$$\mathbb{E}||f^* - \widehat{g}||^2_{\mathcal{P}} = O\Big(\frac{||f^*||_{\mathcal{K}} \log N}{\sqrt{N}}\Big). \tag{9}$$

## 3.3 Remark about randomized spaces

Note that the bounds on the excess risk obtained in (7), (8), and (9) do not depend on the distribution $\mathcal{P}$ under which the data are generated. This is crucial in our setting since $\mathcal{P}$ is usually unknown. It should be noticed that this property does not hold when one considers non-randomized approximation spaces. Indeed, it is relatively easy to exhibit a particularly well-chosen set of features $\varphi_i$ that will approximate functions in a given class using a particular measure $\mathcal{P}$. For example when $\mathcal{P} = \lambda$, the Lebesgue measure, and $f^* \in H^s([0,1]^d)$ (with $s > d/2$), then linear regression using wavelets (with at least $d/2$ vanishing moments), which form an orthonormal basis of $L_{2,\lambda}([0,1]^d)$, enables to achieve a bound similar to (8). However, this is no more the case when $\mathcal{P}$ is not the Lebesgue measure and it seems difficult to modify the features $\varphi_i$ in order to recover the same bound, even when $\mathcal{P}$ is known. This seems to be even harder when $\mathcal{P}$ is arbitrary and not known in advance.

Randomization enables to define approximation spaces such that the approximation error (either in expectation or in high probability on the choice of the random space) is controlled, whatever the measure $\mathcal{P}$ used to assess the performance (even when $\mathcal{P}$ is unknown) is.

For illustration, consider a very peaky (a spot) distribution $\mathcal{P}$ in a high-dimensional space $\mathcal{X}$. Regular linear approximation, say with wavelets (see e.g. [6]), will most probably miss the specific characteristics of $f^*$ at the spot, since the first wavelets have large support. On the contrary, scrambled wavelets, which are functions that contain (random combinations of) all wavelets, will be able to detect correlations between the data and some high frequency wavelets, and thus discover relevant features of $f^*$ at the spot. This is illustrated in the numerical experiment below.

Here $\mathcal{P}$ is a very peaky Gaussian distribution and $f^*$ is a 1-dimensional periodic function. We consider as initial features $(\varphi_i)_{i \geq 1}$ the set of hat functions defined in Section 3.1. Figure 3.3 shows the target function $f^*$, the distribution $\mathcal{P}$, and the data $(x_n, y_n)_{1 \leq n \leq 100}$ (left plots). The middle plots represents the least-squares estimate $\widehat{g}$ using $P = 40$ scrambled objects $(\psi_p)_{1 \leq p \leq 40}$ (here Brownian motions). The right plots shows the least-squares estimate using the initial features $(\varphi_i)_{1 \leq i \leq 40}$. The top figures represent a high level view of the whole domain $[0,1]$. No method is able to learn $f^*$ on the whole space (this is normal since the available data are only generated from a peaky distribution). The bottom figures shows a zoom $[0.45, 0.51]$ around the data. Least-squares regression using scrambled objects is able to learn the structure of $f^*$ in terms of the measure $\mathcal{P}$.

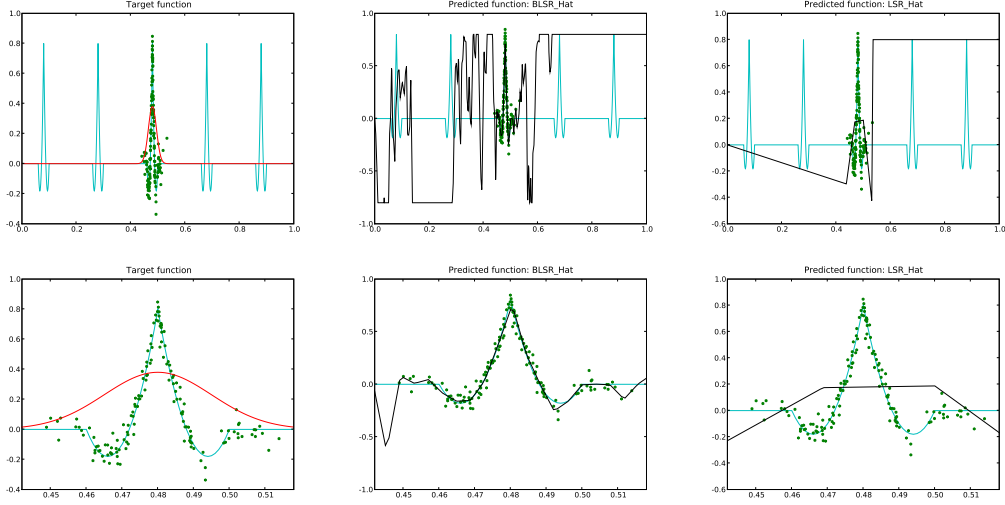

Figure 1: LS estimate of $f^*$ using $N = 100$ data generated from a peaky distribution $\mathcal{P}$ (left plots), using 40 Brownian motions ($\psi_p$) (middle plots) and 40 hat functions ($\varphi_i$) (right plots). The bottom row shows a zoom around the data.

## 4   Discussion

**Minimax optimality:**   Note that although the rate $\tilde{O}(N^{-1/2})$ deduced in (9), does not depend on the dimension $d$ of the input data $\mathcal{X}$, it does not contradict the known minimax lower bounds, which are $\Omega(N^{-2s/(2s+d)})$ for functions defined over $[0,1]^d$ that possess $s$-degrees of smoothness (e.g. that are $s$-times differentiable), see e.g. Chapter 3 of [7]. Indeed, the kernel space $\mathcal{K}$ is composed of functions whose order of smoothness may depend on $d$. For illustration, in the case of scrambled wavelets, the kernel space is the Sobolev space $H^s([0,1]^d)$ with $s > d/2$. Thus $2s/(2s+d) > 1/2$.

Notice that if one considers wavelets with $q$ vanishing moments, where $q > d/2$, then one may choose $s$ (such that $q > s > d/2$) arbitrarily close to $d/2$, and deduce that the excess risk rate $\tilde{O}(N^{-1/2})$ deduced from Theorem 2 is arbitrarily close to the minimax lower rate. Thus regression using scrambled wavelets is minimax optimal (up to logarithmic factors).

Now, concerning Brownian sheets, we are not aware of minimax lower bounds for Cameron-Martin spaces, thus we do not know whether regression using Brownian sheets is minimax optimal or not.

**Links with RKHS Theory:**   There are strong links between the kernel space of Gaussian objects (see eq.(4)) and Reproducing Kernel Hilbert Spaces (RKHS). We now remind two properties that illustrate those links:

- Kernel spaces of Gaussian objects can be built using a Carleman operator, i.e. a linear injective mapping $J : \mathcal{H} \mapsto \mathcal{S}$ (where $\mathcal{H}$ is a Hilbert space) such that $J(h)(t) = \int \Gamma_t(s)h(s)ds$ where $(\Gamma_t)_t$ is a collection of functions of $\mathcal{H}$. There is a bijection between Carleman operators and the set of RKHSs [4, 15].

- Expansion of a Mercer kernel. The expansion of a Mercer kernel $k$ (i.e. when $\mathcal{X}$ is compact Haussdorff and $k$ is a continuous kernel) is given by $k(x,y) = \sum_{i=1}^{\infty} \lambda_i e_i(x)e_i(y)$, where $(\lambda_i)_i$ and $(e_i)_i$ are the eigenvalues and eigenfunctions of the integral operator $L_k : L_{2,\mu}(\mathcal{X}) \to L_{2,\mu}(\mathcal{X})$ defined by $(L_k(f))(x) = \int_{\mathcal{X}} k(x,y)f(y)d\mu(y)$. The associated RKHS is $\mathcal{K} = \{f = \sum_i \alpha_i \varphi_i; \ \sum_i \alpha_i^2 < \infty\}$, where $\varphi_i = \sqrt{\lambda_i}e_i$, endowed with the inner product $\langle f_\alpha, f_\beta \rangle = \langle \alpha, \beta \rangle_{l_2}$. This space is thus also the kernel space of the Gaussian object as defined by (4).

The expansion of a Mercer kernel gives an explicit construction of the functions of the RKHS. However it may not be straightforward to compute the eigenvalues and eigenfunctions of the integral operator $L_k$ and thus the basis functions $\varphi_i$ in the general case.

The approach described in this paper enables to choose explicitly the initial basis functions, and build the corresponding kernel space. For example we have presented examples of expansions using multi-resolution bases (such as hat functions and wavelets), which is not easy to obtain from the Mercer expansion. This is interesting because from the choice of the initial basis, we can characterize the corresponding approximations spaces (e.g. Sobolev space in the case of wavelets). Another more practical benefit is that by using multi-resolution bases (with compact mother function), we can derive efficient numerical implementations, as described in Section 5.

**Related works** In [14, 13], the authors consider, for a given parameterized function $\Phi : \mathcal{X} \times \Theta \to \mathbb{R}$ bounded by 1, and a probability measure $\mu$ over $\Theta$, the space $\mathcal{F}$ of functions $f(x) = \int_\Theta \alpha(\theta)\Phi(x,\theta)d\theta$ such that $||f||_\mu = \sup_\theta |\frac{\alpha(\theta)}{\mu(\theta)}| < \infty$. They show that this is a dense subset of the RKHS with kernel $k(x,y) = \int_\Theta \mu(\theta)\Phi(x,\theta)\Phi(y,\theta)d\theta$, and that if $f \in \mathcal{F}$, then with high probability over $(\theta_p)_{p \leq P} \overset{i.i.d}{\sim} \mu$, there exist coefficients $(c_p)_{p \leq P}$ such that $\widehat{f}(x) = \sum_{p=1}^P c_p \Phi(x,\theta_p)$ satisfies $||\widehat{f} - f||_2^2 \leq O(\frac{||f||_\mu}{\sqrt{P}})$. The method is analogous to the construction of the empirical estimates $g_{A\alpha} \in \mathcal{G}_P$ of function $f_\alpha \in \mathcal{K}$ in our setting. Indeed we may formally identify $\Phi(x,\theta_p)$ with $\psi_p(x) = \sum_i A_{p,i}\varphi_i(x)$, $\theta_p$ with the sequence $(A_{p,i})_i$, and the law $\mu$ with the law of this infinite sequence. However, in our setting we do not require the condition $\sup_{x,\theta} \Phi(x,\theta) \leq 1$ to hold and the fact that $\Theta$ is a set of infinite sequences makes the identification tedious without the Gaussian random functions theory used here. Anyway, we believe that this link provides a better mutual understanding of both approaches (i.e. [14] and this paper).

In the work [1], the authors provide excess risk bounds for greedy algorithms (i.e. in a non-linear approximation setting). The bounds derived in their Theorem 3.1 is similar to the result stated in our Theorem 2. The main difference is that their bound makes use of the $l_1$ norm of the coefficients $\alpha^*$ instead of the $l_2$ norm in our setting. It would be interesting to further investigate whether this difference is a consequence of the non-linear aspect of their approximation or if it results from the different assumptions made about the approximation spaces, in terms of rate of decrease of the coefficients.

## 5  Efficient implementation using a lazy multi-resolution expansion

In practice, in order to build the least-squares estimate, one needs to compute the values of the random features $(\psi_p)_{1 \leq p \leq P}$ at the data points $(x_n)_{1 \leq n \leq N}$, i.e. the matrix $\Psi = (\psi_p(x_n))_{p \leq P, n \leq N}$.

Due to finite memory and precision of computers, numerical implementations can only handle a finite number $F$ of initial features $(\varphi_i)_{1 \leq i \leq F}$. In [10] it was mentioned that the computation of $\Psi$, which makes use of the random matrix $A = (A_{p,i})_{p \leq P, i \leq F}$, has a complexity $O(FPN)$. However, in the multi-resolution schemes described here, provided that the mother function has compact support (such as the hat functions or the Daubechie wavelets), we can significantly speed up the computation of the matrix $\Psi$ by using a *tree-based lazy expansion*, i.e. where the expansion of the random features $(\psi_p)_{p \leq P}$ is built only when needed for the evaluation at the points $(x_n)_n$.

Consider the example of the scrambled wavelets. In dimension 1, using a wavelet dyadic-tree of depth $H$ (i.e. $F = 2^{H+1}$), the numerical cost for computing $\Psi$ is $O(HPN)$ (using one tree per random feature). Now, in dimension $d$ the classical extension of one-dimensional wavelets uses a family of $2^d - 1$ wavelets, thus requires $2^d - 1$ trees each one having $2^{dH}$ nodes. While the resulting number of initial features $F$ is of order $2^{d(H+1)}$, thanks to the lazy evaluation (notice that one never computes all the initial features), one needs to expand at most one path of length $H$ per training point, and the resulting complexity to compute $\Psi$ is $O(2^d HPN)$.

Note that one may alternatively use the so-called sparse-grids instead of wavelet trees, which have been introduced by Griebel and Zenger (see [18, 3]). The main result is that one can reduce significantly the total number of features to $F = O(2^H H^d)$ (while preserving a good approximation for sufficiently smooth functions). Similar lazy evaluation techniques can be applied to sparse-grids.

Now, using a finite $F$ introduces an additional approximation (squared) error term in the final excess risk bounds or order $O(F^{-\frac{2s}{d}})$ for a wavelet basis adapted to $H^s([0,1]^d)$. This additional error (due to the numerical approximation) can be made arbitrarily small, e.g. $o(N^{-1/2})$, whenever $H \geq \frac{\log N}{d}$.

Thus, using $P = O(\sqrt{N})$ random features, we deduce that the complexity of building the matrix $\Psi$ is $O(2^d N^{3/2} \log N)$. Then in order to solve the least squares system, one has to compute $\Psi^T \Psi$, that has numerical cost $O(P^2 N)$, and then solve the system by inversion, which has numerical cost $O(P^{2.376})$ by [5]. Thus, the overall cost of the algorithm is $O(2^d N^{3/2} \log N + N^2)$.

## 6 Conclusion and future works

We analyzed least-squares regression using sub-spaces $\mathcal{G}_P$ that are generated by $P$ random linear combinations of infinitely many initial features. We showed that the approximation space $\mathcal{K} = \{f_\alpha, ||\alpha|| < \infty\}$ (which is also the kernel space of the related Gaussian object) provides a characterization of the set of target functions $f^*$ for which this random regression works. We illustrated the approach on two examples for which the approximation space is a known functional space, namely a Cameron-Martin space when the random features are Brownian sheets (generated by random combinations at all scales of a hat function), and a Sobolev space in the case of scrambled wavelets. We derived a general approximation error result from which we deduced excess risk bounds of order $O(\frac{\log N}{N} P + \frac{\log N}{P} ||f^*||_{\mathcal{K}}^2)$.

We showed that least-squares regression with scrambled wavelets provides rates that are arbitrarily close to minimax optimality. However in the case of regression with Brownian sheets, we are not aware of minimax lower bounds for Cameron-Martin spaces in dimension $d > 1$.

We discussed a key aspect of randomized approximation spaces which is that the approximation error can be controlled independently of the measure $\mathcal{P}$ used to assess the performance. This is essential in a regression setting where $\mathcal{P}$ is unknown, and excess risk rates independent of $\mathcal{P}$ are obtained.

We concluded by mentioning a nice property of using multiscale objects like Brownian sheets and scrambled wavelets (with compact mother wavelet) which is the possibility to be efficiently implemented. We described a lazy expansion approach for computing the regression function which has a numerical complexity $O(N^2 + 2^d N^{3/2} \log N)$.

A limitation of the current scrambled wavelets is that, so far, we did not consider refined analysis for spaces $H^s$ with large smoothness $s \gg d/2$. Possible directions for better handling such spaces may involve refined covering number bounds which will be the object of future works.

## Acknowledgment

This work has been supported by French National Research Agency (ANR) through COSINUS program (project EXPLO-RA number ANR-08-COSI-004).

## Footnotes

[1]In the full rank case when $N \geq P$, $\Psi^\dagger = (\Psi^T \Psi)^{-1} \Psi^T$

# References

[1] Andrew Barron, Albert Cohen, Wolfgang Dahmen, and Ronald Devore. Approximation and learning by greedy algorithms. 36:1:64–94, 2008.

[2] Gerard Bourdaud. Ondelettes et espaces de besov. *Rev. Mat. Iberoamericana*, 11:3:477–512, 1995.

[3] Hans-Joachim Bungartz and Michael Griebel. Sparse grids. In Arieh Iserles, editor, *Acta Numerica*, volume 13. University of Cambridge, 2004.

[4] Stéphane Canu, Xavier Mary, and Alain Rakotomamonjy. Functional learning through kernel. *arXiv*, 2009, October.

[5] D. Coppersmith and S. Winograd. Matrix multiplication via arithmetic progressions. In *STOC '87: Proceedings of the nineteenth annual ACM symposium on Theory of computing*, pages 1–6, New York, NY, USA, 1987. ACM.

[6] R. DeVore. *Nonlinear Approximation*. Acta Numerica, 1997.

[7] L. Györfi, M. Kohler, A. Krzyżak, and H. Walk. *A distribution-free theory of nonparametric regression*. Springer-Verlag, 2002.

[8] Stéphane Jaffard. Décompositions en ondelettes. In *Development of mathematics 1950–2000*, pages 609–634. Birkhäuser, Basel, 2000.

[9] Svante Janson. *Gaussian Hilbert spaces*. Cambridge Univerity Press, Cambridge, UK, 1997.

[10] Odalric-Ambrym Maillard and Rémi Munos. Compressed Least-Squares Regression. In *NIPS 2009*, Vancouver Canada, 2009.

[11] Odalric-Ambrym Maillard and Rémi Munos. Linear regression with random projections. Technical report, Hal INRIA: http://hal.archives-ouvertes.fr/inria-00483014/, 2010.

[12] Stephane Mallat. *A Wavelet Tour of Signal Processing*. Academic Press, 1999.

[13] Ali Rahimi and Benjamin Recht. Random features for large-scale kernel machines. In John C. Platt, Daphne Koller, Yoram Singer, Sam T. Roweis, John C. Platt, Daphne Koller, Yoram Singer, and Sam T. Roweis, editors, *NIPS*. MIT Press, 2007.

[14] Ali Rahimi and Benjamin Recht. Uniform approximation of functions with random bases. 2008.

[15] S. Saitoh. *Theory of reproducing Kernels and its applications*. Longman Scientific & Technical, Harlow, UK, 1988.

[16] Robert Tibshirani. Regression shrinkage and selection via the Lasso. *Journal of the Royal Statistical Society, Series B*, 58:267–288, 1994.

[17] A. N. Tikhonov. Solution of incorrectly formulated problems and the regularization method. *Soviet Math Dokl 4*, pages 1035–1038, 1963.

[18] C. Zenger. Sparse grids. In W. Hackbusch, editor, *Parallel Algorithms for Partial Differential Equations, Proceedings of the Sixth GAMM-Seminar*, volume 31 of Notes on Num. Fluid Mech., Kiel, 1990. Vieweg-Verlag.

